# How Perception Guides Production in Birdsong Learning

**Christopher L. Fry**
cfry@cogsci.ucsd.edu
Department of Cognitive Science
University of California at San Diego
La Jolla, CA 92093-0515

## Abstract

A computational model of song learning in the song sparrow (*Melospiza melodia*) learns to categorize the different syllables of a song sparrow song and uses this categorization to train itself to reproduce song. The model fills a crucial gap in the computational explanation of birdsong learning by exploring the organization of perception in songbirds. It shows how competitive learning may lead to the organization of a specific nucleus in the bird brain, replicates the song production results of a previous model (Doya and Sejnowski, 1995), and demonstrates how perceptual learning can guide production through reinforcement learning.

## 1 INTRODUCTION

The *passeriformes* or songbirds make up more than half of all bird species and are divided into two groups: the *oscines* which learn their songs and *sub-oscines* which do not. *Oscines* raised in isolation sing degraded species typical songs similar to wild song. Deafened *oscines* sing completely degraded songs (Konishi, 1965) , while deafened *sub-oscines* develop normal songs (Kroodsma and Konishi, 1991) indicating that auditory feedback is crucial in *oscine* song learning.

Innate structures in the bird brain regulate song learning. For example, song sparrows show innate preferences for their own species' songs and song structure (Marler, 1991). Innate preferences are thought to be encoded in an **auditory template** which limits the sounds young birds may copy. According to the **auditory template hypothesis** birds go through two phases during song learning, a **memorization phase** and a **motor phase**. In the **memorization phase**, which lasts from approximately 20 to 50 days after birth in the song sparrow, the bird selects which sounds to copy based on an innate template and refines the template based

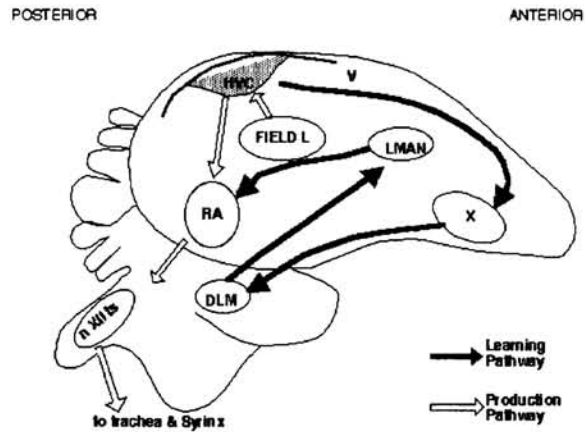

Figure 1: A simplified sketch of a saggital section of the songbird brain. **Field L** (Field L) receives auditory input and projects to the production pathway: **HVc** (formerly the caudal nucleus of the hyperstriatum), **RA** (robust nucleus of archistriatum), **nXIIts** (hypoglossal nerve), the syrinx (vocal organ) and the learning pathway: **X** (area X), **DLM** (medial nucleus of the dorsolateral thalamus), **LMAN** (lateral magnocellular nucleus of the anterior neostriatum), **RA** (Konishi, 1989; Vicario, 1994). **V** is the lateral ventricle.

on the sounds it hears. In the **motor phase** (from approximately 272 to 334 days after birth) the template provides feedback during singing. Learning to sing the memorized, template song is a gradual process of refining the produced song to match memory (Marler, 1991).

A song is made up of phrases, phrases of syllables and syllables of notes. Syllables, usually separated by periods of silence, are the main units of analysis. Notes typically last from 10-100 msecs and are used to construct syllables (100-200 msecs) which are reused to produce trills and other phrases.

## 2 NEUROBIOLOGY OF SONG

The two main neural pathways that govern song are the motor and learning pathways seen in figure 1 (Konishi, 1989). Lesions to the motor pathway interrupt singing throughout life while lesions to the learning pathway disrupt early song learning. Although these pathways seem to have segregated functions, recordings of neurons during song playback have shown that cells throughout the song system respond to song (Konishi, 1989).

Studies of song perception have shown the best auditory stimulus that will evoke a response in the song system is the bird's own song (Margoliash, 1986). The song specific neurons in **HVc** of the white-crowned sparrow often require a sequence of two syllables to respond (Margoliash, 1986; Margoliash and Fortune, 1992) and are made up of two main types in **HVc**. One type is sensitive to temporal combinations of stimuli while the other is sensitive to harmonic characteristics (Margoliash and Fortune, 1992).

## 3 COMPUTATION

Previous computational work on birdsong learning predicted individual neural responses using back-propagation (Margoliash and Bankes, 1993) and modelled motor mappings for song production (Doya and Sejnowski, 1995). The current work de-

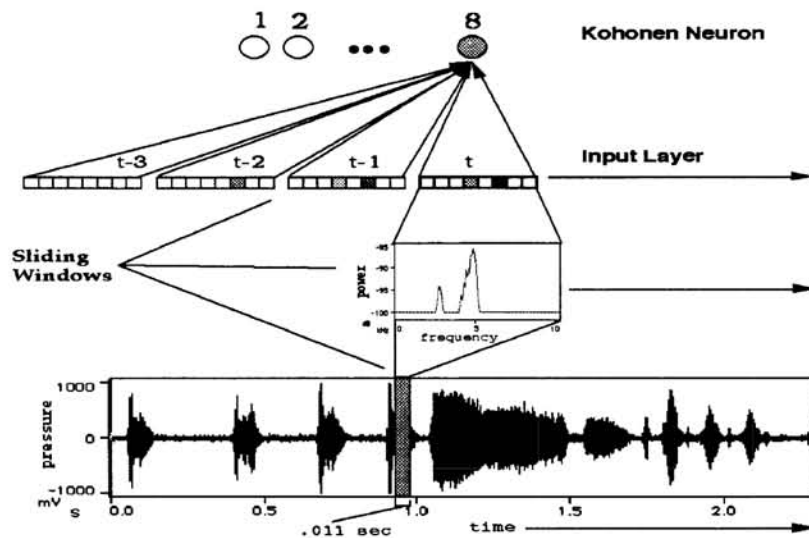

Figure 2: Perceptual network input encoding. The song is converted into frequency bins which are presented to the Kohonen layer over four time steps.

velops a model of birdsong syllable perception which extends Doya and Sejnowski's (1995) model of birdsong learning. Birdsong syllable segmentation is accomplished using an unsupervised system and this system is used to train the network to reproduce its input using reinforcement learning.

The model implements the two phases of the auditory template hypothesis, **memorization** and **motor**. In the first phase the template song is segmented into syllables by an unsupervised Kohonen network (Kohonen, 1984). In the second phase the syllables are reproduced using a reinforcement learning paradigm based on Doya and Sejnowski (1995).

The model extends previous work in three ways: 1) a self-organizing network picks out syllables in the song; 2) the self-organizing network provides feedback during song production; and 3) a more biologically plausible model of the syrinx is used to generate song.

## 3.1 Perception

Recognizing a syllable involves identifying a short sequence of notes. Kohonen networks use an unsupervised learning method to categorize an input space based on similar neural responses. Thus a Kohonen network is a natural candidate for identifying the syllables in a song.

One song from the repertoire of a song sparrow was chosen as the training song for the network. The song was encoded by passing a sliding window across the training waveform (sampled at 22.255 kHz) of the selected song. At each time step, a non-overlapping 256 point ($\approx$ .011 sec) fast fourier transform (FFT) was used to generate a power spectrum (figure 2). The power spectrum was divided into 8 bins. Each bin was mapped to a real number using a gaussian summation procedure with the peak of the gaussian at the center of each frequency bin. Four time-steps were passed to each Kohonen neuron.

The network's task was to identify similar syllables in the input song. The input song was broken down into syllables by looking for points where the power at all

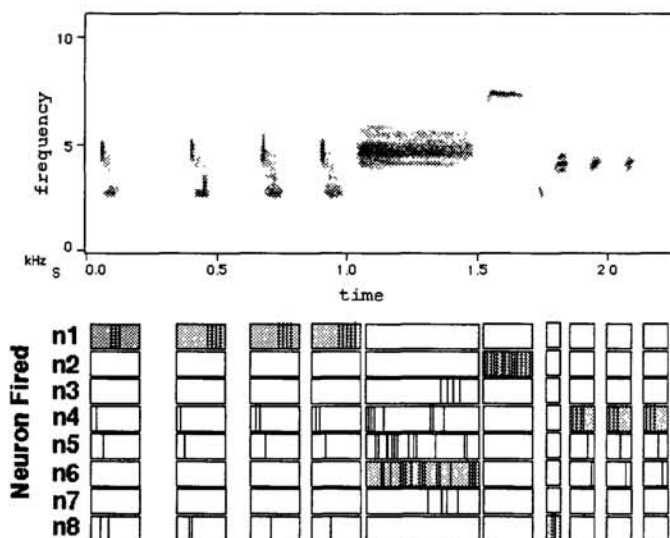

Figure 3: Categorization of song syllables by a Kohonen network. The power-spectrum of the training song is at the top. The responses of the Kohonen neurons are at the bottom. For each time-step the winning neuron is shown with a vertical bar. The shaded areas indicate the neuron that fired the most during the presentation of the syllable.

frequencies dropped below a threshold. A syllable was defined as sound of duration greater than .011 seconds bounded by two low-power points. The network was not trained on the noise between syllables. The song was played for the network ten times (1050 training vectors), long enough for a stable response pattern to emerge.

The activation of a neuron was: $Net_j = \Sigma x_i w_{ij}$. Where: $Net_j$ = output of neuron $j$, $w_{ij}$ = the weight connecting $input_i$ to $neuron_j$, $x_i = input_i$. The Kohonen network was trained by initializing the connection weights to $1/\sqrt{number\ of\ neurons}$ + small random component ($r \leq .01$), normalizing the inputs, and updating the weights to the winning neuron by the following rule: $\mathbf{w}_{new} = \mathbf{w}_{old} + \alpha(\mathbf{x} - \mathbf{w}_{old})$ where: $\alpha = training\ rate = .20$. If the same neuron won twice in a row the training rate was decreased by $1/2$. Only the winning neuron was reinforced resulting in a non-localized feature map.

### 3.1.1   Perceptual Results

The Kohonen network was able to assign a unique neuron to each type of syllable (figure 3). Of the eight neurons in the network, the one that fired the most frequently during the presentation of a syllable uniquely identified the type of syllable. The first four syllables of the input song sound alike, contain similar frequencies, and are coded by the first neuron (**N1**). The last three syllables sound alike, contain similar frequencies, and are coded by the fourth neuron (**N4**). Syllable five was coded by neuron six (**N6**), syllable six by neuron two (**N2**) and syllable seven by neuron eight (**N8**).

Figure 4 shows the frequency sensitivity of each neuron (1-8, figure 3) plotted against each time step (1-4). This plot shows the harmonic and temporally sensitive neurons that developed during the learning phase of the Kohonen network. Neuron 2 is sensitive to only one frequency at approximately 6-7 kHz, indicated by the solid white band across the 6-7 kHz frequency range in figure 4. Neuron 4 is sensitive to mid-range frequencies of short duration. Note that in figure 4 **N4** responds

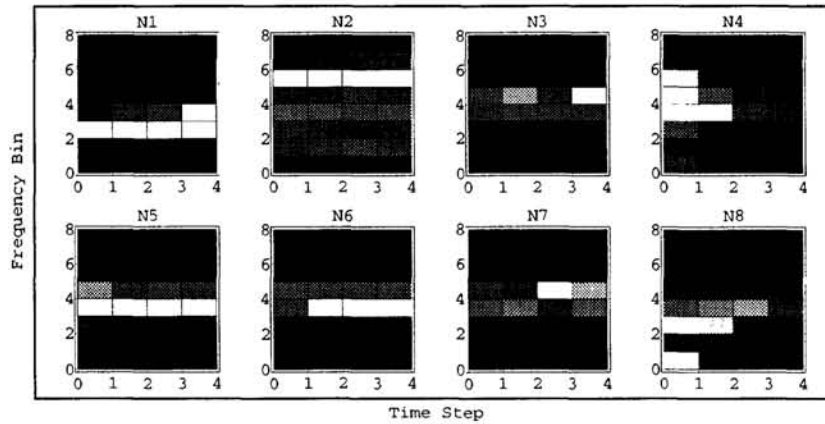

Figure 4: The values of the weights mapping frequency bins and time steps to Kohonen neurons. White is maximum, Black is minimum.

maximally to mid-range frequencies only in the first two time steps. It uses this temporal sensitivity to distinguish between the last three syllables and the fifth syllable (figure 3) by keying off the length of time mid-range frequencies are present. Contrast this early response sensitivity with neuron 6, which is sensitive to mid-range frequencies of long duration, but responds only *after* one time step. It uses this temporal sensitivity to respond to the long sustained frequency of syllable four. Considered together, neurons 2,4,6 and 8 illustrate the two types of neurons (temporal and harmonic) found in **HVc** by Margoliash and Fortune (1993). Competitive learning may underly the formation of these neurons in **HVc**.

### 3.2 Production

After competitive learning trains the perceptual part of the network to categorize the song into syllables, the perceptual network can be used to train the production side of the network to sing.

The first step in modelling song production is to create a model of the avian vocal apparatus, the syrinx. In the syrinx sounds arise when air flows through the syringeal passage and causes the tympanic membrane to vibrate. The frequency is controlled by the tension of the membrane controlled by the syringeal musculature. The amplitude is dependent on the area of the syringeal orifice which is dependent on the tension of the labium. The interactions of this system were modelled by modulated sine waves. Four parameters governed the fundamental frequency(**p**), frequency modulation(**tm**), amplitude (**ex**) and frequency of amplitude modulation(**l**). The range of the parameters was set according to calculations in Greenwalt (1968). The parameters were combined in the following equation (based on Greenwalt, 1968), $f(ex, l, p, tm, t) = \mathbf{ex}\cos(2\pi t\, \mathbf{l})\cos(2\pi t\,\mathbf{p} + cos(2\pi t\,\mathbf{tm}))$.

Using this equation song can be generated over time by making assumptions about the response properties of neurons in **RA**. Following Doya and Sejnowski (1995) it was assumed that pools of **RA** neurons have different temporal response profiles. Syllable like temporal responses can be generated by modifying the weights from the Kohonen layer (**HVc**) to the production layer (**RA**).

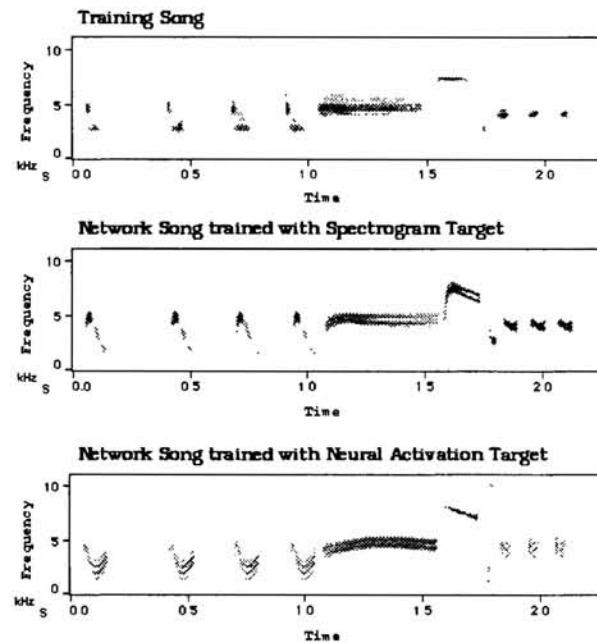

Figure 5: Training song and two songs produced with different representations of the training song.

The production side of the network was trained using the reinforcement learning paradigm described in Doya and Sejnowski (1995). Each syllable was presented in the order it occurred in the training song to the Kohonen layer, which turned on a single neuron. A random vector was added to the weights from the Kohonen layer to the output layer and a syllable was produced. The produced syllable was compared to the stored representation of the template song which was used to generate an error signal and an estimate of the gradient. If the evaluation of the produced syllable was better than a threshold the weights were kept, otherwise they were discarded.

Two experiments were done using different representations of the template song. In the first experiment the template song was the stored power spectrum of each syllable and the error signal was the cosine of the angle between the power spectrum of the produced syllable and the template syllable. In the second experiment the template song was the stored neural responses to song (recorded during the memorization phase) and the error signal was the Euclidean distance between neural responses to the produced syllable and the neural responses to the template song.

### 3.2.1 Production Results

Figure 5 shows the output of the production network after training with different representations of the training song. The network was able to replicate the major frequency components of the training song to a high degree of accuracy. The song trained with the spectrogram target was learned to a 90% average cosine between the spectrograms of the produced song and the training song on each syllable with the best syllable learned to 100% accuracy and the worst to 85% after 1000 trials. A crucial aspect to achieving performance was smoothing the template spectrogram. The third song shows that the network was able to learn the template song using the neural responses of the perceptual system to generate the reinforcement signal. The average distance between the initial randomly produced syllables and the training

song was reduced by 50%.

## 4   DISCUSSION

This work fills a crucial gap in the computational explanation of song learning left by prior work. Doya and Sejnowski (1995) showed how song could be produced but left unanswered the questions of how song is perceived and how the perceptual system provides feedback during song production. This study shows a time-delay Kohonen network can learn to categorize the syllables of a sample song and this network can train song production with no external teacher. The Kohonen network explains how neurons sensitive to temporal and harmonic structure could arise in the songbird brain through competitive learning. Taken as a whole, the model presents a concrete proposal of the computational principles governing the **Auditory Template Hypothesis** and how a song is memorized and used to train song production. Future work will flesh out the effects of innate structure on learning by examining how the settings of the initial weights on the network affect song learning and predict experimental effects of deafening and isolation.

### Acknowledgements

Thanks to S. Vehrencamp for providing the song data, J. Batali, J. Elman, J. Bradbury and T. Sejnowski for helpful comments, and K. Doya for advice on replicating his model.

### References

Doya, K. and Sejnowski, T.J. (1995). A novel reinforcement model of birdsong vocalization learning. In Tesauro, G., Touretzky, D. S. and Leen, T.K., editors, *Advances in Neural Information Processing Systems 7*. MIT Press, Cambridge, MA.

Greenwalt, C.H. (1968). Bird Song: Acoustics and Physiology. Smithsonian Institution Press. Wash., D.C.

Kohonen, T. (1984). *Self-organization and Associative Memory, Vol. 8*. Springer-Verlag, Berlin.

Konishi, M. (1965). The role of auditory feedback in the control of vocalization in the white-crowned sparrow. *Zeitschrift fur Tierpsychogie*, 22,770-783.

Konishi, M. (1989). Birdsong for Neurobiologists. *Neuron*, 3, 541-549.

Kroodsma, D.E. and Konishi, M. (1991). A suboscine bird (eastern phoebe, Sayonoris phoebe) develops normal song without auditory feedback. *Animal Behavior*, 42, 477-487.

Marler, P. (1991). The instinct to learn. In *The Epigenesis of Mind: Essays on Biology and Cognition*, eds. S. Carey and R. Gelman. Lawrence Erlbaum Associates.

Margoliash, D. (1986). Preference for autogenous song by auditory neurons in a song system nucleus of the white-crowned sparrow. *Journal of Neuroscience*, 6,1643-1661.

Margoliash, D. and Bankes, S.C. (1993). Computations in the Ascending Auditory Pathway in Songbirds Related to Song Learning. *American Zoologist*, 33, 94-103.

Margoliash, D. and Fortune, E. (1992). Temporal and Harmonic Combination-Sensitive Neurons in the Zebra Finch's HVc. *Journal of Neuroscience*, 12, 4309-4326.

Vicario, D. (1994). Motor Mechanisms Relevant to Auditory-Vocal Interactions in Songbirds. *Brain, Behavior and Evolution*,44, 265-278.